# TRAINING A 3-NODE NEURAL NETWORK IS NP-COMPLETE

Avrim Blum[*]
MIT Lab. for Computer Science
Cambridge, Mass. 02139 USA

Ronald L. Rivest[†]
MIT Lab. for Computer Science
Cambridge, Mass. 02139 USA

## ABSTRACT

We consider a 2-layer, 3-node, $n$-input neural network whose nodes compute linear threshold functions of their inputs. We show that it is NP-complete to decide whether there exist weights and thresholds for the three nodes of this network so that it will produce output consistent with a given set of training examples. We extend the result to other simple networks. This result suggests that those looking for perfect training algorithms cannot escape inherent computational difficulties just by considering only simple or very regular networks. It also suggests the importance, given a training problem, of finding an appropriate network and input encoding for that problem. It is left as an open problem to extend our result to nodes with non-linear functions such as sigmoids.

## INTRODUCTION

One reason for the recent surge in interest in neural networks is the development of the "back-propagation" algorithm for training neural networks. The ability to train large multi-layer neural networks is essential for utilizing neural networks in practice, and the back-propagation algorithm promises just that. In practice, however, the back-propagation algorithm runs very slowly, and the question naturally arises as to whether there are necessarily intrinsic computational difficulties associated with training neural networks, or whether better training algorithms might exist. This paper provides additional support for the position that training neural networks is intrinsically difficult.

A common method of demonstrating a problem to be intrinsically hard is to show the problem to be "NP-complete". The theory of NP-complete problems is well-understood (Garey and Johnson, 1979), and many infamous problems—such as the traveling salesman problem—are now known to be NP-complete. While NP-completeness does not render a problem totally unapproachable in

---
[*]Supported by an NSF graduate fellowship.

[†]This paper was prepared with support from NSF grant DCR-8607494, ARO Grant DAAL03-86-K-0171, and the Siemens Corporation.

practice, it usually implies that only small instances of the problem can be solved exactly, and that large instances can at best only be solved approximately, even with large amounts of computer time.

The work in this paper is inspired by Judd (Judd, 1987) who shows the following problem to be NP-complete:

> "Given a neural network and a set of training examples, does there exist a set of edge weights for the network so that the network produces the correct output for all the training examples?"

Judd also shows that the problem remains NP-complete even if it is only required a network produce the correct output for two-thirds of the training examples, which implies that even approximately training a neural network is intrinsically difficult in the worst case. Judd produces a class of networks and training examples for those networks such that any training algorithm will perform poorly on some networks and training examples in that class. The results, however, do not specify any particular "hard network"—that is, any single network hard for all algorithms. Also, the networks produced have a number of hidden nodes that grows with the number of inputs, as well as a quite irregular connection pattern.

We extend his result by showing that it is NP-complete to train a specific very simple network, having only two hidden nodes and a regular interconnection pattern. We also present classes of regular 2-layer networks such that for *all* networks in these classes, the training problem is hard in the worst case (in that there exists some hard sets of training examples). The NP-completeness proof also yields results showing that finding approximation algorithms that make only one-sided error or that approximate closely the minimum number of hidden-layer nodes needed for a network to be powerful enough to correctly classify the training data, is probably hard, in that these problems can be related to other difficult (but not known to be NP-complete) approximation problems.

Our results, like Judd's, are described in terms of "batch"-style learning algorithms that are given all the training examples at once. It is worth noting that training with an "incremental" algorithm that sees the examples one at a time such as back-propagation is at least as hard. Thus the NP-completeness result given here also implies that incremental training algorithms are likely to run slowly.

Our results state that given a network of the classes considered, for any training algorithm there will be some types of training problems such that the algorithm will perform poorly as the problem size increases. The results leave open the possibility that given a training problem that is hard for some network, there might exist a different network and encoding of the input that make training easy.

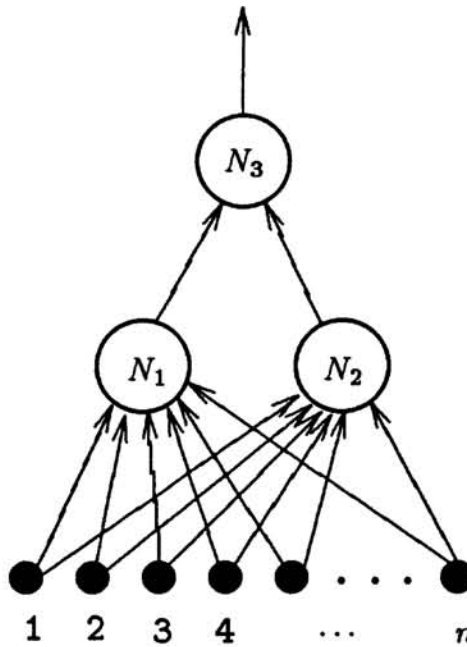

Figure 1: The three node neural network.

# THE NEURAL NETWORK TRAINING PROBLEM

The multilayer network that we consider has $n$ binary inputs and three nodes: $N_1, N_2, N_3$. All the inputs are connected to nodes $N_1$ and $N_2$. The outputs of hidden nodes $N_1$ and $N_2$ are connected to output node $N_3$ which gives the output of the network.

Each node $N_i$ computes a linear threshold function $f_i$ on its inputs. If $N_i$ has input $x = (x_1, \ldots, x_m)$, then for some constants $a_0, \ldots, a_m$,

$$f_i(x) = \begin{cases} +1 & \text{if } a_1 x_1 + a_2 x_2 + \cdots + a_m x_m > a_0 \\ -1 & \text{otherwise.} \end{cases}$$

The $a_j$'s $(j \geq 1)$ are typically viewed as weights on the incoming edges and $a_0$ as the threshold.

The training algorithm for the network is given a set of training examples. Each is either a positive example (an input for which the desired network output is $+1$) or a negative example (an input for which the desired output is $-1$). Consider the following problem. Note that we have stated it as a decision ("yes" or "no") problem, but that the search problem (finding the weights) is at least equally hard.

TRAINING A 3-NODE NEURAL NETWORK:

**Given:** A set of $O(n)$ training examples on $n$ inputs.

**Question:** Do there exist linear threshold functions $f_1, f_2, f_3$ for nodes $N_1, N_2, N_3$

such that the network of figure 1 produces outputs consistent with the training set?

**Theorem:**  *Training a 3-node neural network is NP-complete.*

We also show (proofs omitted here due to space requirements) NP-completeness results for the following networks:

1. The 3-node network described above, even if any or all of the weights for one hidden node are required to equal the corresponding weights of the other, so possibly only the thresholds differ, and even if any or all of the weights are forced to be from $\{+1, -1\}$.

2. Any $k$-hidden node, for $k$ bounded by some polynomial in $n$ (eg: $k = n^2$), two-layer fully-connected network with linear threshold function nodes where the output node is required to compute the AND function of its inputs.

3. The 2-layer, 3-node $n$-input network with an XOR output node, if ternary features are allowed.

In addition we show (proof omitted here) that any set of positive and negative training examples classifiable by the 3-node network with XOR output node (for which training is NP-complete) can be correctly classified by a perceptron with $O(n^2)$ inputs which consist of the original $n$ inputs and all products of pairs of the original $n$ inputs (for which training can be done in polynomial-time using linear programming techniques).

## THE GEOMETRIC POINT OF VIEW

A training example can be thought of as a point in $n$-dimensional space, labeled '+' or '−' depending on whether it is a positive or negative example. The points are vertices of the $n$-dimensional hypercube. The zeros of the functions $f_1$ and $f_2$ for the hidden nodes can be thought of as $(n-1)$-dimensional hyperplanes in this space. The planes $P_1$ and $P_2$ corresponding to the functions $f_1$ and $f_2$ divide the space into four quadrants according to the four possible pairs of outputs for nodes $N_1$ and $N_2$. If the planes are parallel, then one or two of the quadrants is degenerate (non-existent). Since the output node receives as input only the outputs of the hidden nodes $N_1$ and $N_2$, it can only distinguish between points in different quadrants. The output node is also restricted to be a linear function. It may not, for example, output "+1" when its inputs are $(+1, +1)$ and $(-1, -1)$, and output "−1" when its inputs are $(+1, -1)$ and $(-1, +1)$.

So, we may reduce our question to the following: given $O(n)$ points in $\{0, 1\}^n$, each point labeled '+' or '−', does there exist either

1.  a single plane that separates the '+' points from the '−' points, or

2.  two planes that partition the points so that either one quadrant contains all and only '+' points or one quadrant contains all and only '−' points.

We first look at the restricted question of whether there exist two planes that partition the points such that one quadrant contains all and only the '+' points. This corresponds to having an "AND" function at the output node. We will call this problem: "2-Linear Confinement of Positive Boolean Examples". Once we have shown this to be NP-complete, we will extend the proof to the full problem by adding examples that disallow the other possibilities at the output node.

Megiddo (Megiddo, 1986) has shown that for $O(n)$ *arbitrary* '+' and '−' points in $n$-dimensional Euclidean space, the problem of whether there exist two hyperplanes that separate them is NP-complete. His proof breaks down, however, when one restricts the coordinate values to $\{0, 1\}$ as we do here. Our proof turns out to be of a quite different style.

## SET SPLITTING

The following problem was proven to be NP-complete by Lovasz (Garey and Johnson 1979).

SET-SPLITTING:

**Given:** A finite set $S$ and a collection $C$ of subsets $c_i$ of $S$.

**Question:** Do there exist disjoint sets $S_1$, $S_2$ such that $S_1 \cup S_2 = S$ and $\forall i, c_i \not\subset S_1$ and $c_i \not\subset S_2$.

The Set-Splitting problem is also known as 2-non-Monotone Colorability. Our use of this problem is inspired by its use by Kearns, Li, Pitt, and Valiant to show that learning k-term DNF is NP-complete (Kearns et al. 1987) and the style of the reduction is similar.

## THE REDUCTION

Suppose we are given an instance of the Set-Splitting problem:

$$S = \{s_i\}, \ C = \{c_j\}, \ c_j \subseteq S, \ |S| = n.$$

Create the following signed points on the $n$-dimensional hypercube $\{0, 1\}^n$:

*   Let the origin $0^n$ be labeled '+'.

*   For each $s_i$, put a point labeled '−' at the neighbor to the origin that has a 1 in the $i$th bit—that is, at $(\overset{1\,2\ \cdots\ i\ \cdots\ n}{00\cdots010\cdots0})$. Call this point $p_i$.

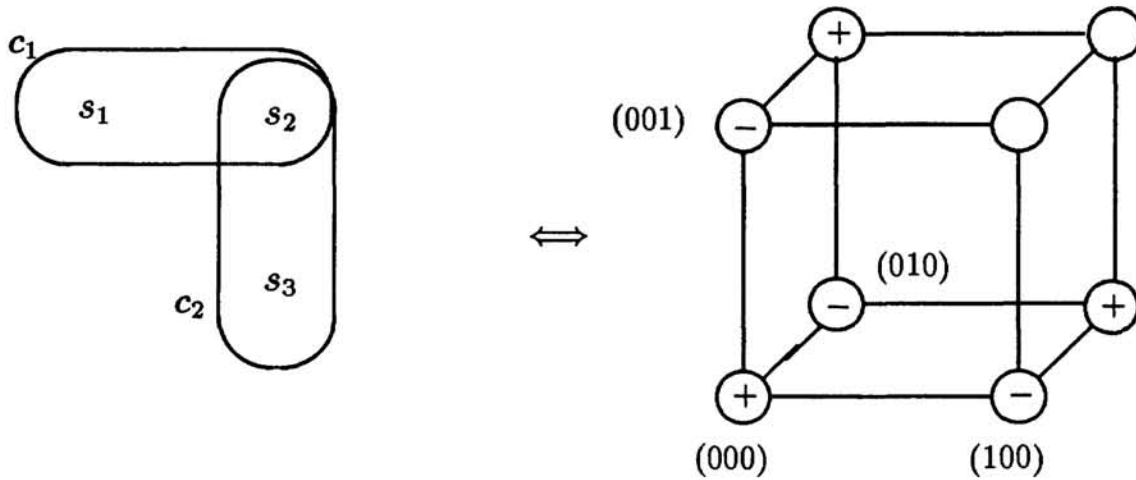

Figure 2: An example.

- For each $c_j = \{s_{j1}, \ldots, s_{jk_j}\}$, put a point labeled '+' at the location whose bits are 1 at exactly the positions $j_1, j_2, \ldots, j_{k_j}$—that is, at $p_{j1} + \cdots + p_{jk_j}$.

For example, let $S = \{s_1, s_2, s_3\}$, $C = \{c_1, c_2\}$, $c_1 = \{s_1, s_2\}$, $c_2 = \{s_2, s_3\}$.

So, we create '−' points at: (0 0 1), (0 1 0), (1 0 0)

and '+' points at: (0 0 0), (1 1 0), (0 1 1) in this reduction (see figure 2).

**Claim:** The given instance of the Set-Splitting problem has a solution $\Longleftrightarrow$ the constructed instance of the 2-Linear Confinement of Positive Boolean Examples problem has a solution.

**Proof:** $(\Rightarrow)$

Given $S_1$ from the solution to the Set-Splitting instance, create the plane $P_1$ : $a_1 x_1 + \cdots + a_n x_n = -\frac{1}{2}$, where $a_i = -1$ if $s_i \in S_1$, and $a_i = n$ if $s_i \notin S_1$. Let the vectors $a = (a_1, \ldots a_n), x = (x_1, \ldots, x_n)$.

This plane separates from the origin exactly the '−' points corresponding to $s_i \in S_1$ and no '+' points. Notice that for each $s_i \in S_1$, $a \cdot p_i = -1$, and that for each $s_i \notin S_1$, $a \cdot p_i = n$. For each '+' point $p$, $a \cdot p > -\frac{1}{2}$ since either $p$ is the origin or else $p$ has a 1 in a bit $i$ such that $s_i \notin S_1$.

Similarly, create the plane $P_2$ from $S_2$.

$(\Leftarrow)$

Let $S_1$ be the set of points separated from the origin by $P_1$ and $S_2$ be those points separated by $P_2$. Place any points separated by both planes in either $S_1$ or $S_2$ arbitrarily. Sets $S_1$ and $S_2$ cover $S$ since all '−' points are separated from the origin by at least one of the planes. Consider some $c_j = \{s_{j1} \cdots s_{jk_j}\}$

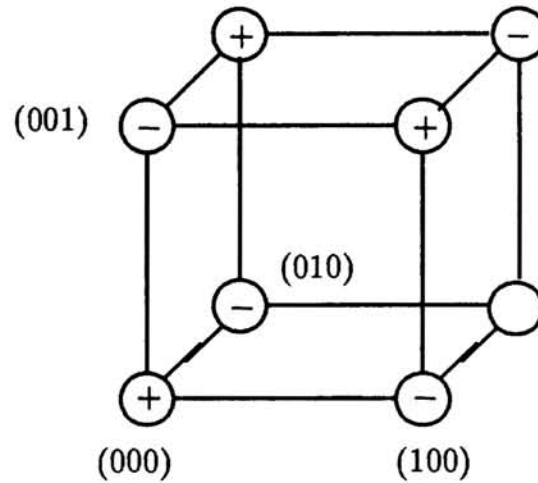

Figure 3: The gadget.

and the corresponding '−' points $p_{j1}, \ldots, p_{jk_j}$. If, say, $c_j \subset S_1$, then $P_1$ must separate all the $p_{ji}$ from the origin. Therefore, $P_1$ must separate $p_{j1} + \cdots + p_{jk_j}$ from the origin. Since that point is the '+' point corresponding to $c_j$, the '+' points are not all confined to one quadrant, contradicting our assumptions. So, no $c_j$ can be contained in $S_1$. Similarly, no $c_j$ can be contained in $S_2$. ∎

We now add a "gadget" consisting of 6 new points to handle the other possibilities at the output node. The gadget forces that the only way in which two planes could linearly separate the '+' points from the '−' points would be to confine the '+' points to one quadrant. The gadget consists of extra points and three new dimensions. We add three new dimensions, $x_{n+1}, x_{n+2}$, and $x_{n+3}$, and put '+' points in locations:

$$(0 \cdots 0 \; 101), \quad (0 \cdots 0 \; 011)$$

and '−' points in locations:

$$(0 \cdots 0 \; 100), \quad (0 \cdots 0 \; 010), \quad (0 \cdots 0 \; 001), \quad (0 \cdots 0 \; 111).$$

(See figure 3.)

The '+' points of this cube can be separated from the '−' points by appropriate settings of the weights of planes $P_1$ and $P_2$ corresponding to the three new dimensions. Given planes $P_1' : a_1 x_1 + \cdots + a_n x_n = -\frac{1}{2}$ and $P_2' : b_1 x_1 + \cdots + b_n x_n = -\frac{1}{2}$ which solve a 2-Linear Confinement of Positive Boolean Examples instance in $n$ dimensions, expand the solution to handle the gadget by setting

$$P_1 \quad \text{to} \quad a_1 x_1 + \cdots + a_n x_n + x_{n+1} + x_{n+2} - x_{n+3} = -\frac{1}{2}$$

$$P_2 \quad \text{to} \quad b_1 x_1 + \cdots + b_n x_n - x_{n+1} - x_{n+2} + x_{n+3} = -\frac{1}{2}$$

($P_1$ separates '−' point $(0 \cdots 0\ 001)$ from the '+' points and $P_2$ separates the other three '−' points from the '+' points). Also, notice that there is no way in which just one plane can separate the '+' points from the '−' points in the cube and also no way for two planes to confine all the negative points in one quadrant. Thus we have proved the theorem.

## CONCLUSIONS

Training a 3-node neural network whose nodes compute linear threshold functions is NP-complete.

An open problem is whether the NP-completeness result can be extended to neural networks that use sigmoid functions. We believe that it can because the use of sigmoid functions does not seem to alter significantly the expressive power of a neural network. Note that Judd (Judd 1987), for the networks he considers, shows NP-completeness for a wide variety of node functions including sigmoids.

**References**

James A. Anderson and Edward Rosenfeld, editors. *Neurocomputing: Foundations of Research*. MIT Press, 1988.

M. Garey and D. Johnson. *Computers and Intractability: A Guide to the Theory of NP-Completeness*. W. H. Freeman, San Francisco, 1979.

J. Stephen Judd. Learning in networks is hard. In *Proceedings of the First International Conference on Neural Networks*, pages 685–692, I.E.E.E., San Diego, California June 1987.

J. Stephen Judd. *Neural Network Design and the Complexity of Learning*. PhD thesis, Computer and Information Science dept., University of Massachusetts, Amherst, Mass. U.S.A., 1988.

Michael Kearns, Ming Li, Leonard Pitt, and Leslie Valiant. On the learnability of boolean formulae. In *Proceedings of the Nineteenth Annual ACM Symposium on Theory of Computing*, pages 285–295, New York, New York, May 1987.

Nimrod Megiddo. *On The Complexity of Polyhedral Separability*. Technical Report RJ 5252, IBM Almaden Research Center, August 1986.

Marvin Minsky and Seymour Papert. *Perceptrons: An Introduction to Computational Geometry*. The MIT Press, 1969.

David E. Rumelhart and James L. McClelland, editors. *Parallel Distributed Processing (Volume I: Foundations)*. MIT Press, 1986.
